# DISTRIBUTED NEURAL INFORMATION PROCESSING
# IN THE VESTIBULO-OCULAR SYSTEM

Clifford Lau
Office of Naval Research Detachment
Pasadena, CA 91106

Vicente Honrubia*
UCLA Division of Head and Neck Surgery
Los Angeles, CA 90024

## ABSTRACT

A new distributed neural information-processing model is proposed to explain the response characteristics of the vestibulo-ocular system and to reflect more accurately the latest anatomical and neurophysiological data on the vestibular afferent fibers and vestibular nuclei. In this model, head motion is sensed topographically by hair cells in the semicircular canals. Hair cell signals are then processed by multiple synapses in the primary afferent neurons which exhibit a continuum of varying dynamics. The model is an application of the concept of "multilayered" neural networks to the description of findings in the bullfrog vestibular nerve, and allows us to formulate mathematically the behavior of an assembly of neurons whose physiological characteristics vary according to their anatomical properties.

## INTRODUCTION

Traditionally the physiological properties of individual vestibular afferent neurons have been modeled as a linear time-invariant system based on Steinhausen's description of cupular motion.[1] The vestibular nerve input to different parts of the central nervous system is usually represented by vestibular primary afferents that have

response properties defined by population averages from individual neurons.[2]

A new model of vestibular nerve organization is proposed to account for the observed variabilities in the primary vestibular afferent's anatomical and physiological characteristics. The model is an application of the concept of "multilayered" neural networks,[3,4] and it attempts to describe the behavior of the entire assembly of vestibular neurons based on new physiological and anatomical findings in the frog vestibular nerve. It was found that primary vestibular afferents show systematic differences in sensitivity and dynamics and that there is a correspondence between the individual neuron's physiological properties and the location of innervation in the area of the crista and also the sizes of the neuron's fibers and somas. This new view of topological organization of the receptor and vestibular nerve afferents is not included in previous models of vestibular nerve function. Detailed findings from this laboratory on the anatomical and physiological properties of the vestibular afferents in the bullfrog have been published.[5,6]

## REVIEW OF THE ANATOMY AND PHYSIOLOGY OF THE VESTIBULAR NERVE

The most pertinent anatomical and physiological data on the bullfrog vestibular afferents are summarized here. In the vestibular nerve from the anterior canal four major branches (bundles) innervate different parts of the crista (Figure 1). From serial histological sections it has been shown that fibers in the central bundle innervate hair cells at the center of the crista, and the lateral bundles project to the periphery of the crista. In each nerve there is an average of $1170 \pm 171$ (n = 5) fibers, of which the thick fibers (diameter > 7.0 microns, large dots) constitute 8% and the thin fibers (< 4.0 microns, small dots) 76%. The remaining fibers (16%) fall into the range between 4.0 and 7.0 microns. We found that the thick fibers innervate only the center of the crista, and the thinner ones predominantly innervate the periphery.

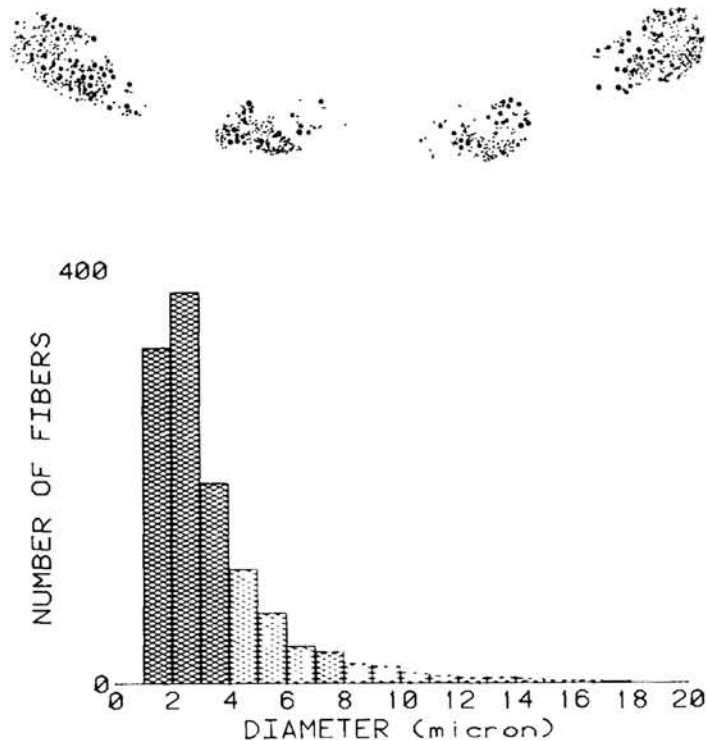

400

Fig. 1. Number of fibers and their diameters in the anterior semicircular canal nerve in the bullfrog.

There appears to be a physiological and anatomical correlation between fiber size and degree of regularity of spontaneous activity. By recording from individual neurons and subsequently labeling them with horseradish peroxidase intracellularly placed in the axon, it is possible to visualize and measure individual ganglion cells and axons and to determine the origin of the fiber in the crista as well as the projections in different parts of the vestibular nuclei. Figure 2 shows an example of three neurons of different sizes and degrees of regularity of spontaneous activity. In general, fibers with large diameters tend to be more irregular with large coefficients of variation (CV) of the interspike intervals, whereas thin fibers tend to be more regular. There is also a relationship for each neuron between CV and the magnitude of the response to physiological rotatory stimuli, that is, the response gain. (Gain is defined as the ratio of the response in spikes per second to the stimulus in degrees per second.) Figure 3 shows a plot of gain as a function of CV as well as of fiber diameter. For the more regular fibers (CV < 0.5), the gain tends to increase as the diameter of the fiber increases.

460

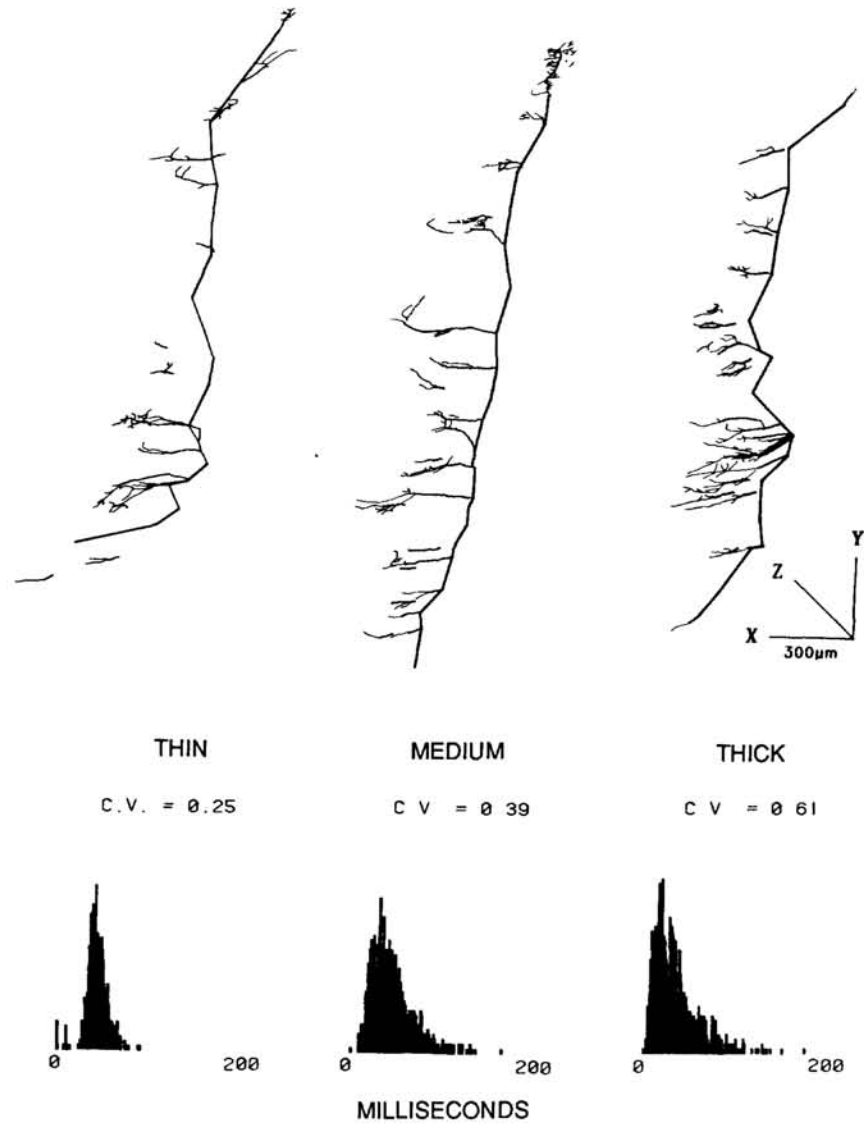

THIN

C.V. = 0.25

MEDIUM

C V = 0 39

THICK

C V = 0 61

0    200        0    200        0    200

MILLISECONDS

Fig. 2. Examples of thin, medium and thick fibers and their spontaneous activity. CV - coefficient of variation.

For the more irregular fibers (CV > 0.5), the gain tends to remain the same with increasing fiber diameter (4.9 ± 1.9 spikes/second/degrees/second).

Figure 4 shows the location of projection of the afferent fibers at the vestibular nuclei from the anterior, posterior, and horizontal canals and saccule. There is an overall organization in the pattern of innervation from the afferents of each vestibular organ to the vestibular nuclei, with fibers from different receptors overlapping in various

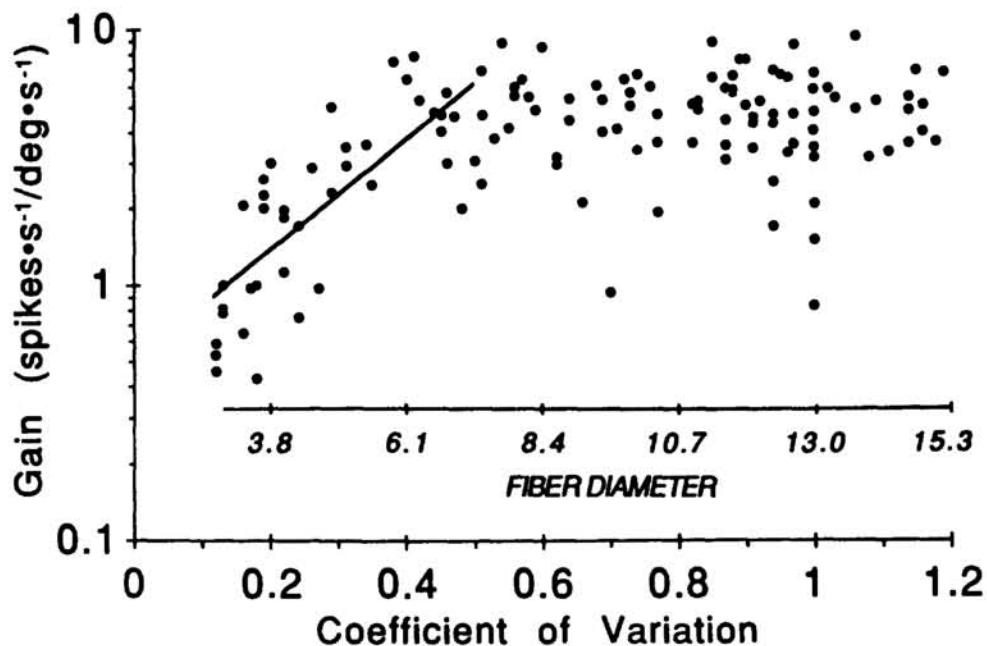

Fig. 3. Gain versus fiber diameters and CV. Stimulus was a sinusoidal rotation of 0.05 Hz at 22 degrees/second peak velocity.

parts of the vestibular nuclei. Fibers from the anterior semicircular canal tend to travel ventrally, from the horizontal canal dorsally, and from the posterior canal the most dorsally.

For each canal nerve the thick fibers (indicated by large dots) tend to group together to travel lateral to the thin fibers (indicated by diffused shading); thus, the topographical segregation between thick and thin fibers at the periphery is preserved at the vestibular nuclei.

In following the trajectories of individual neurons in the central nervous system, however, we found that each fiber innervates all parts of the vestibular nuclei, caudally to rostrally as well as transversely, and because of the spread of the large number of branches, as many as 200 from each neuron, there is a great deal of overlap among the projections.

## DISTRIBUTED NEURAL INFORMATION-PROCESSING MODEL

Figure 5 represents a conceptual organization, based on the above anatomical and physiological data, of Scarpa's

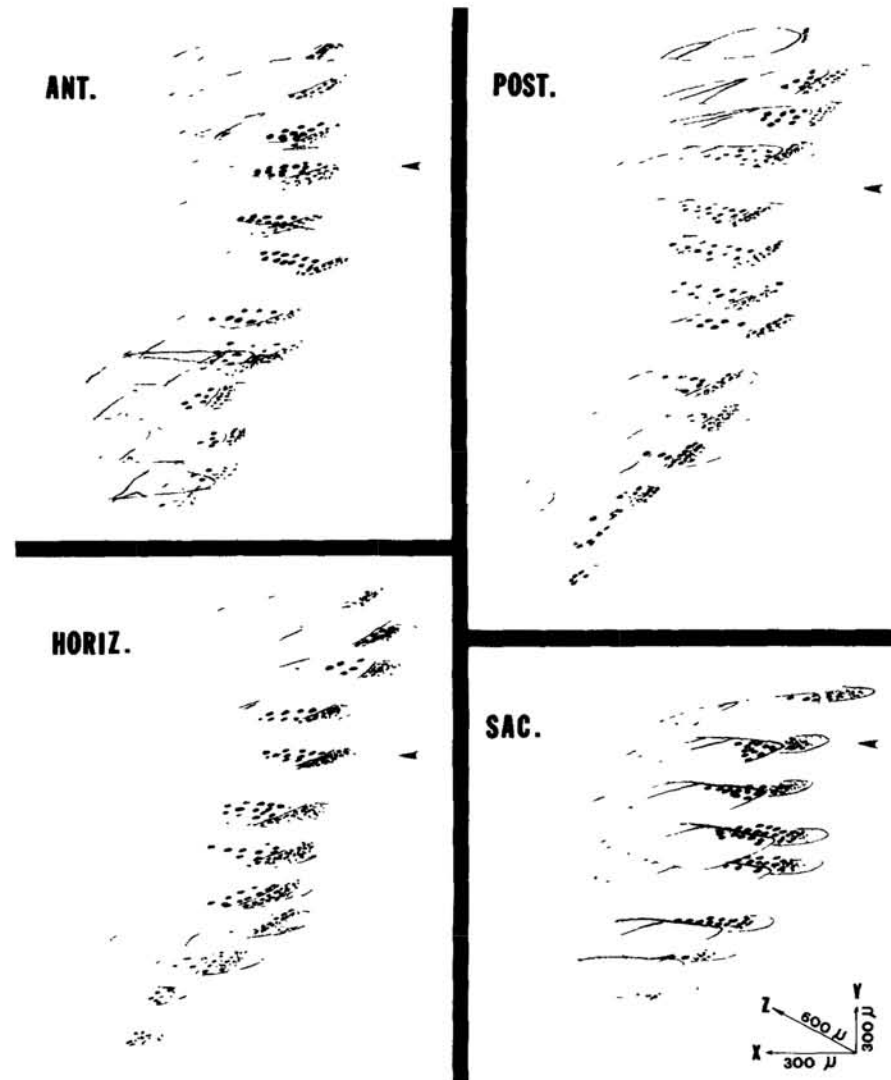

Fig. 4. Three-dimensional reconstruction of the primary afferent fibers' location in the vestibular nuclei.

ganglion cells of the vestibular nerve and their innervation of the hair cells and of the vestibular nuclei. The diagram depicts large Scarpa's ganglion cells with thick fibers innervating restricted areas of hair cells near the center of the crista (top) and smaller Scarpa's ganglion cells with thin fibers on the periphery of the crista innervating multiple hair cells with a great deal of overlap among fibers. At the vestibular nuclei, both thick and thin fibers innervate large areas with a certain gradient of overlapping among fibers of different diameters.

The new distributed neural information-processing model for the vestibular system is based on this anatomical organization, as shown in Figure 6. The response

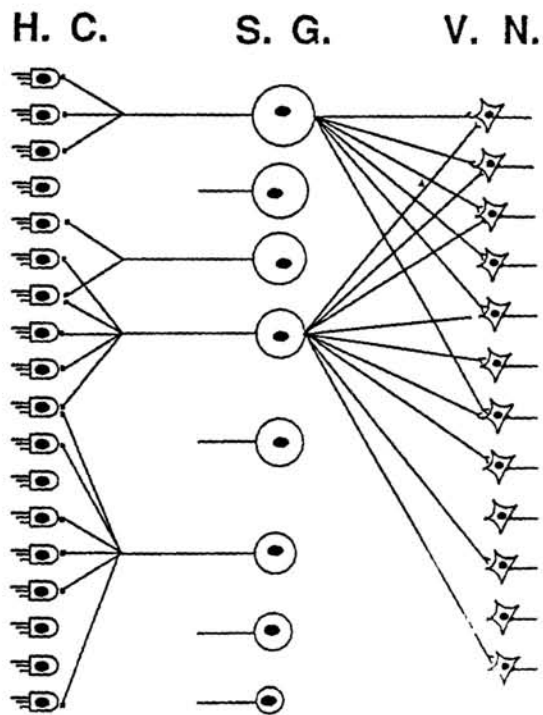

Fig. 5. Anatomical organization of the vestibular nerve. H.C. - hair cells. S.G. - Scarpa's ganglion cells. V.N. - vestibular nuclei.

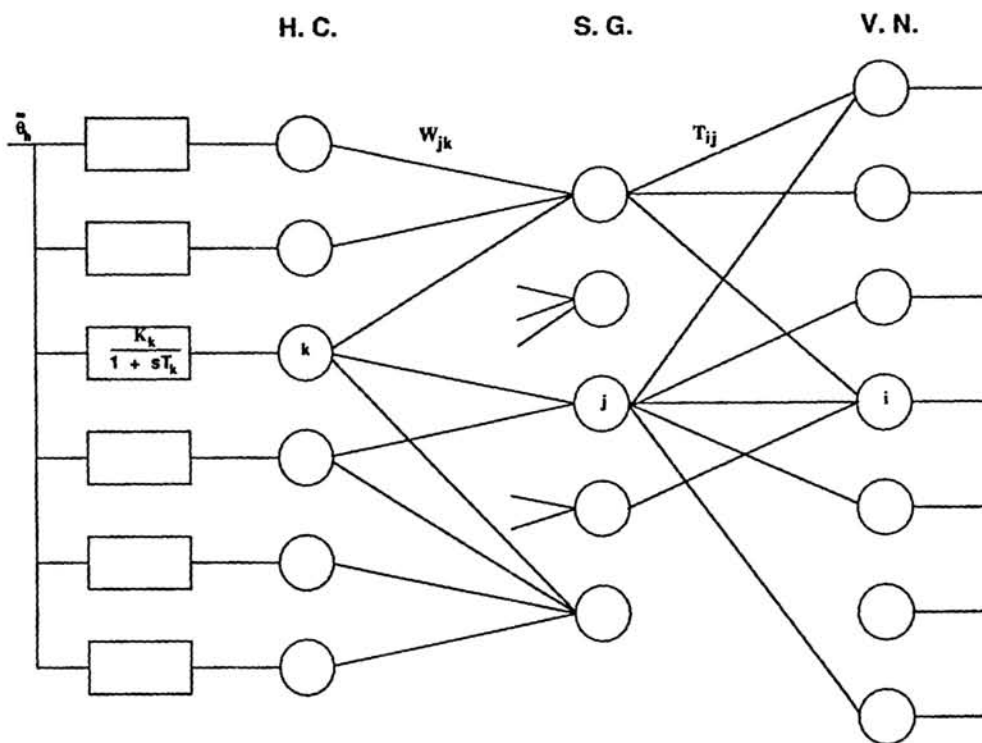

Fig. 6. Distributed neural information-processing model of the vestibular nerve.

characteristic of the primary afferent fiber is represented by the transfer function $SG_j(s)$. This transfer function serves as a description of the gain and phase response of individual neurons to angular rotation. The simplest model would be a first-order system with d.c. gain $K_j$ (spikes/ second over head acceleration) and a time constant $T_j$ (seconds) for the $jth$ fiber as shown in equation (1):

$$SG_j(s) = \frac{K_j}{1 + sT_j}.\qquad(1)$$

For the bullfrog, $K_j$ can range from about 3 to 25 spikes/second/degree/second$^2$, and $T_j$ from about 10 to 0.5 second. The large and high-gain neurons are more phasic than the small neurons and tend to have shorter time constants. As described above, $K_j$ and $T_j$ for the $jth$ neuron are functions of location and fiber diameter. Bode plots (gain and phase versus frequency) of experimental data seem to indicate, however, that a better transfer function would consist of a higher-order system that includes fractional power. This is not surprising since the afferent fiber response characteristic must be the weighted sum of several electromechanical steps of transduction in the hair cells. A plausible description of these processes is given in equation (2):

$$SG_j(s) = \sum_k W_{jk} \frac{K_k}{1 + sT_k},\qquad(2)$$

where gain $K_k$ and time constant $T_k$ are the electro- mechanical properties of the hair cell-cupula complex and are functions of location on the crista, and $W_{jk}$ is the synaptic efficacy (strength) between the $jth$ neuron and the $kth$ hair cell. In this context, the transfer function given in equation (1) provides a measure of the "weighted average" response of the multiple synapses given in equation (2).

We also postulate that the responses of the vestibular nuclei neurons reflect the weighted sums of the responses of the primary vestibular afferents, as follows:

$$VN_i = f \left( \sum_j T_{ij} \, SG_j \right) , \tag{3}$$

where f(.) is a sigmoid function describing the change in firing rates of individual neurons due to physiological stimulation. It is assumed to saturate between 100 to 300 spikes/second, depending on the neuron. $T_{ij}$ is the synaptic efficacy (strength) between the *ith* vestibular neuron and the *jth* afferent fiber.

## CONCLUSIONS

Based on anatomical and physiological data from the bullfrog we presented a description of the organization of the primary afferent vestibular fibers. The responses of the afferent fibers represent the result of summated excitatory processes. The information on head movement in the assemblage of neurons is codified as a continuum of varying physiological responses that reflect a sensoritopic organization of inputs from the receptor to the central nervous system. We postulated a new view of the organization in the peripheral vestibular organs and in the vestibular nuclei. This view does not require unnecessary simplification of the varying properties of the individual neurons. The model is capable of extracting the weighted average response from assemblies of large groups of neurons while the unitary contribution of individual neurons is preserved. The model offers the opportunity to incorporate further developments in the evaluation of the different roles of primary afferents in vestibular function. Large neurons with high sensitivity and high velocity of propagation are more effective in activating reflexes that require quick responses such as vestibulo-spinal and vestibulo-ocular reflexes. Small neurons with high thresholds for the generation of action potentials and lower sensitivity are more tuned to the maintenance of posture

and muscle tonus. We believe the physiological differences reflect the different physiological roles.

In this emerging scheme of vestibular nerve organization it appears that information about head movement, topographically filtered in the crista, is distributed through multiple synapses in the vestibular centers. Consequently, there is also reason to believe that different neurons in the vestibular nuclei preserve the variability in response characteristics and the topological discrimination observed in the vestibular nerve. Whether this idea of the organization and function of the vestibular system is valid remains to be proven experimentally.

## Footnotes

*Work supported by grants NS09823 and NS08335 from the National Institutes of Health (NINCDS) and grants from the Pauley Foundation and the Hope for Hearing Research Foundation.

## REFERENCES

1. W. Steinhausen, Arch. Ges. Physiol. 217, 747 (1927).
2. J. M. Goldberg and C. Fernandez, in: Handbook of Physiology, Sect. 1, Vol. III, Part 2 (I. Darian-Smith, ed., Amer. Physiol. Soc., Bethesda, MD, 1984), p. 977.
3. D. E. Rumelhart, G. E. Hinton and J. L. McClelland, in: Parallel Distributed Processing: Explorations in the Microstructure of Cognition, Vol. 1: Foundations (D. E. Rumelhart, J. L. McClelland and the PDP Research Group, eds., MIT Press, Cambridge, MA, 1986), p. 45.
4. J. Hopfield, Proc. Natl. Acad. Sci. 79, 2554 (1982).
5. V. Honrubia, S. Sitko, J. Kimm, W. Betts and I. Schwartz, Intern. J. Neurosci. 15, 197 (1981).
6. V. Honrubia, S. Sitko, R. Lee, A. Kuruvilla and I. Schwartz, Laryngoscope 94, 464 (1984).
